# Replicated Softmax: an Undirected Topic Model

**Ruslan Salakhutdinov**
Brain and Cognitive Sciences and CSAIL
Massachusetts Institute of Technology
rsalakhu@mit.edu

**Geoffrey Hinton**
Department of Computer Science
University of Toronto
hinton@cs.toronto.edu

## Abstract

We introduce a two-layer undirected graphical model, called a "Replicated Softmax", that can be used to model and automatically extract low-dimensional latent semantic representations from a large unstructured collection of documents. We present efficient learning and inference algorithms for this model, and show how a Monte-Carlo based method, Annealed Importance Sampling, can be used to produce an accurate estimate of the log-probability the model assigns to test data. This allows us to demonstrate that the proposed model is able to generalize much better compared to Latent Dirichlet Allocation in terms of both the log-probability of held-out documents and the retrieval accuracy.

## 1 Introduction

Probabilistic topic models [2, 9, 6] are often used to analyze and extract semantic topics from large text collections. Many of the existing topic models are based on the assumption that each document is represented as a mixture of topics, where each topic defines a probability distribution over words. The mixing proportions of the topics are document specific, but the probability distribution over words, defined by each topic, is the same across all documents.

All these models can be viewed as graphical models in which latent topic variables have directed connections to observed variables that represent words in a document. One major drawback is that exact inference in these models is intractable, so one has to resort to slow or inaccurate approximations to compute the posterior distribution over topics. A second major drawback, that is shared by all mixture models, is that these models can never make predictions for words that are sharper than the distributions predicted by any of the individual topics. They are unable to capture the essential idea of distributed representations which is that the distributions predicted by individual active features get multiplied together (and renormalized) to give the distribution predicted by a whole set of active features. This allows individual features to be fairly general but their intersection to be much more precise. For example, distributed representations allow the topics "government", "mafia" and "playboy" to combine to give very high probability to a word "Berlusconi" that is not predicted nearly as strongly by each topic alone.

To date, there has been very little work on developing topic models using undirected graphical models. Several authors [4, 17] used two-layer undirected graphical models, called Restricted Boltzmann Machines (RBMs), in which word-count vectors are modeled as a Poisson distribution. While these models are able to produce distributed representations of the input and perform well in terms of retrieval accuracy, they are unable to properly deal with documents of different lengths, which makes learning very unstable and hard. This is perhaps the main reason why these potentially powerful models have not found their application in practice. Directed models, on the other hand, can easily handle unobserved words (by simply ignoring them), which allows them to easily deal with different-sized documents. For undirected models marginalizing over unobserved variables is generally a non-trivial operation, which makes learning far more difficult. Recently, [13] attempted to fix this problem by proposing a Constrained Poisson model that would ensure that the mean Poisson

rates across all words sum up to the length of the document. While the parameter learning has been shown to be stable, the introduced model no longer defines a proper probability distribution over the word counts.

In the next section we introduce a "Replicated Softmax" model. The model can be efficiently trained using Contrastive Divergence, it has a better way of dealing with documents of different lengths, and computing the posterior distribution over the latent topic values is easy. We will also demonstrate that the proposed model is able to generalize much better compared to a popular Bayesian mixture model, Latent Dirichlet Allocation (LDA) [2], in terms of both the log-probability on previously unseen documents and the retrieval accuracy.

## 2  Replicated Softmax: A Generative Model of Word Counts

Consider modeling discrete visible units $\mathbf{v}$ using a restricted Boltzmann machine, that has a two-layer architecture as shown in Fig. 1. Let $\mathbf{v} \in \{1, ..., K\}^D$, where $K$ is the dictionary size and $D$ is the document size, and let $\mathbf{h} \in \{0, 1\}^F$ be binary stochastic hidden topic features. Let $\mathbf{V}$ be a $K \times D$ observed binary matrix with $v_i^k = 1$ if visible unit $i$ takes on $k^{th}$ value. We define the energy of the state $\{\mathbf{V}, \mathbf{h}\}$ as follows:

$$E(\mathbf{V}, \mathbf{h}) = -\sum_{i=1}^{D}\sum_{j=1}^{F}\sum_{k=1}^{K} W_{ij}^k h_j v_i^k - \sum_{i=1}^{D}\sum_{k=1}^{K} v_i^k b_i^k - \sum_{j=1}^{F} h_j a_j, \tag{1}$$

where $\{W, a, b\}$ are the model parameters: $W_{ij}^k$ is a symmetric interaction term between visible unit $i$ that takes on value $k$, and hidden feature $j$, $b_i^k$ is the bias of unit $i$ that takes on value $k$, and $a_j$ is the bias of hidden feature $j$ (see Fig. 1). The probability that the model assigns to a visible binary matrix $\mathbf{V}$ is:

$$P(\mathbf{V}) = \frac{1}{\mathcal{Z}} \sum_{\mathbf{h}} \exp\left(-E(\mathbf{V}, \mathbf{h})\right), \quad \mathcal{Z} = \sum_{\mathbf{V}}\sum_{\mathbf{h}} \exp\left(-E(\mathbf{V}, \mathbf{h})\right), \tag{2}$$

where $\mathcal{Z}$ is known as the partition function or normalizing constant. The conditional distributions are given by softmax and logistic functions:

$$p(v_i^k = 1|\mathbf{h}) = \frac{\exp\left(b_i^k + \sum_{j=1}^{F} h_j W_{ij}^k\right)}{\sum_{q=1}^{K} \exp\left(b_i^q + \sum_{j=1}^{F} h_j W_{ij}^q\right)} \tag{3}$$

$$p(h_j = 1|\mathbf{V}) = \sigma\left(a_j + \sum_{i=1}^{D}\sum_{k=1}^{K} v_i^k W_{ij}^k\right), \tag{4}$$

where $\sigma(x) = 1/(1 + \exp(-x))$ is the logistic function.

Now suppose that for each document we create a separate RBM with as many softmax units as there are words in the document. Assuming we can ignore the order of the words, all of these softmax units can share the same set of weights, connecting them to binary hidden units. Consider a document that contains $D$ words. In this case, we define the energy of the state $\{\mathbf{V}, \mathbf{h}\}$ to be:

$$E(\mathbf{V}, \mathbf{h}) = -\sum_{j=1}^{F}\sum_{k=1}^{K} W_j^k h_j \hat{v}^k - \sum_{k=1}^{K} \hat{v}^k b^k - D\sum_{j=1}^{F} h_j a_j, \tag{5}$$

where $\hat{v}^k = \sum_{i=1}^{D} v_i^k$ denotes the count for the $k^{th}$ word. Observe that the bias terms of the hidden units are scaled up by the length of the document. This scaling is crucial and allows hidden topic units to behave sensibly when dealing with documents of different lengths.

Given a collection of $N$ documents $\{\mathbf{V}_n\}_{n=1}^{N}$, the derivative of the log-likelihood with respect to parameters $W$ takes the form:

$$\frac{1}{N}\sum_{n=1}^{N} \frac{\partial \log P(\mathbf{V_n})}{\partial W_j^k} = \mathrm{E}_{P_{\text{data}}}\left[\hat{v}^k h_j\right] - \mathrm{E}_{P_{\text{Model}}}\left[\hat{v}^k h_j\right],$$

where $\mathrm{E}_{P_{\text{data}}}[\cdot]$ denotes an expectation with respect to the data distribution $P_{\text{data}}(\mathbf{h}, \mathbf{V}) = p(\mathbf{h}|\mathbf{V})P_{\text{data}}(\mathbf{V})$, with $P_{\text{data}}(\mathbf{V}) = \frac{1}{N}\sum_n \delta(\mathbf{V} - \mathbf{V}_n)$ representing the empirical distribution,

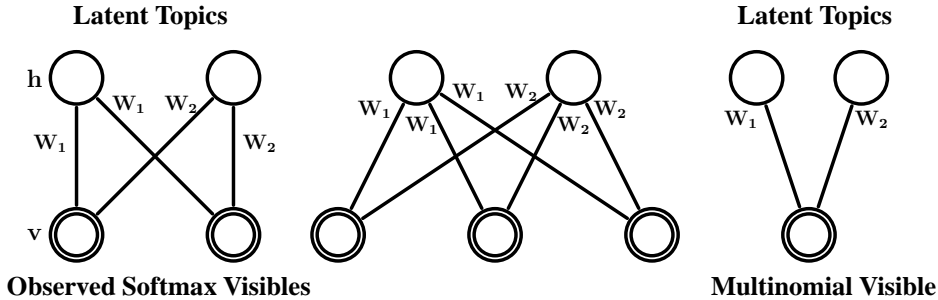

**Latent Topics**                              **Latent Topics**

**Observed Softmax Visibles**               **Multinomial Visible**

Figure 1: Replicated Softmax model. The top layer represents a vector **h** of stochastic, binary topic features and and the bottom layer represents softmax visible units **v**. All visible units share the same set of weights, connecting them to binary hidden units. **Left:** The model for a document containing two and three words. **Right:** A different interpretation of the Replicated Softmax model, in which $D$ softmax units with identical weights are replaced by a single multinomial unit which is sampled $D$ times.

and $E_{P_{\text{Model}}}[\cdot]$ is an expectation with respect to the distribution defined by the model. Exact maximum likelihood learning in this model is intractable because exact computation of the expectation $E_{P_{\text{Model}}}[\cdot]$ takes time that is exponential in $\min\{D, F\}$, i.e the number of visible or hidden units. To avoid computing this expectation, learning is done by following an approximation to the gradient of a different objective function, called the "Contrastive Divergence" (CD) ([7]):

$$\Delta W_j^k \;=\; \alpha\bigg( E_{P_{\text{data}}}\left[\hat{v}^k h_j\right] - E_{P_T}\left[\hat{v}^k h_j\right] \bigg), \tag{6}$$

where $\alpha$ is the learning rate and $P_T$ represents a distribution defined by running the Gibbs chain, initialized at the data, for $T$ full steps. The special bipartite structure of RBM's allows for quite an efficient Gibbs sampler that alternates between sampling the states of the hidden units independently given the states of the visible units, and vise versa (see Eqs. 3, 4). Setting $T = \infty$ recovers maximum likelihood learning.

The weights can now be shared by the whole family of different-sized RBM's that are created for documents of different lengths (see Fig. 1). We call this the "Replicated Softmax" model. A pleasing property of this model is that computing the approximate gradients of the CD objective (Eq. 6) for a document that contains 100 words is computationally not much more expensive than computing the gradients for a document that contains only one word. A key observation is that using $D$ softmax units with identical weights is equivalent to having a single multinomial unit which is sampled $D$ times, as shown in Fig. 1, right panel. If instead of sampling, we use real-valued softmax probabilities multiplied by $D$, we exactly recover the learning algorithm of a Constrained Poisson model [13], except for the scaling of the hidden biases with $D$.

## 3 Evaluating Replicated Softmax as a Generative Model

Assessing the generalization performance of probabilistic topic models plays an important role in model selection. Much of the existing literature, particularly for undirected topic models [4, 17], uses extremely indirect performance measures, such as information retrieval or document classification. More broadly, however, the ability of the model to generalize can be evaluated by computing the probability that the model assigns to the previously unseen documents, which is independent of any specific application.

For undirected models, computing the probability of held-out documents exactly is intractable, since computing the global normalization constant requires enumeration over an exponential number of terms. Evaluating the same probability for directed topic models is also difficult, because there are an exponential number of possible topic assignments for the words.

Recently, [14] showed that a Monte Carlo based method, Annealed Importance Sampling (AIS) [12], can be used to efficiently estimate the partition function of an RBM. We also find AIS attractive because it not only provides a good estimate of the partition function in a reasonable amount of computer time, but it can also just as easily be used to estimate the probability of held-out documents for directed topic models, including Latent Dirichlet Allocation (for details see [16]). This will allow us to properly measure and compare generalization capabilities of Replicated Softmax and

---

**Algorithm 1** Annealed Importance Sampling (AIS) run.

---

1: Initialize $0 = \beta_0 < \beta_1 < ... < \beta_S = 1$.
2: Sample $\mathbf{V}_1$ from $p_0$.
3: **for** $s = 1 : S - 1$ **do**
4:     Sample $\mathbf{V}_{s+1}$ given $\mathbf{V}_s$ using $T_s(\mathbf{V}_{s+1} \leftarrow \mathbf{V}_s)$.
5: **end for**
6: Set $w_{\mathrm{AIS}} = \prod_{s=1}^{S} p_s^*(\mathbf{V}_s)/p_{s-1}^*(\mathbf{V}_s)$.

---

LDA models. We now show how AIS can be used to estimate the partition function of a Replicated Softmax model.

## 3.1 Annealed Importance Sampling

Suppose we have two distributions: $p_A(\mathbf{x}) = p_A^*(\mathbf{x})/\mathcal{Z}_A$ and $p_B(\mathbf{x}) = p_B^*(\mathbf{x})/\mathcal{Z}_B$. Typically $p_A(\mathbf{x})$ is defined to be some simple proposal distribution with known $\mathcal{Z}_A$, whereas $p_B$ represents our complex target distribution of interest. One way of estimating the ratio of normalizing constants is to use a simple importance sampling method:

$$\frac{\mathcal{Z}_B}{\mathcal{Z}_A} = \sum_{\mathbf{x}} \frac{p_B^*(\mathbf{x})}{p_A^*(\mathbf{x})} p_A(\mathbf{x}) = \mathrm{E}_{p_A}\left[\frac{p_B^*(\mathbf{x})}{p_A^*(\mathbf{x})}\right] \approx \frac{1}{N} \sum_{i=1}^{N} \frac{p_B^*(\mathbf{x}^{(i)})}{p_A^*(\mathbf{x}^{(i)})}, \tag{7}$$

where $\mathbf{x}^{(i)} \sim p_A$. However, if the $p_A$ and $p_B$ are not close enough, the estimator will be very poor. In high-dimensional spaces, the variance of the importance sampling estimator will be very large, or possibly infinite, unless $p_A$ is a near-perfect approximation to $p_B$.

Annealed Importance Sampling can be viewed as simple importance sampling defined on a much higher dimensional state space. It uses many auxiliary variables in order to make the proposal distribution $p_A$ be closer to the target distribution $p_B$. AIS starts by defining a sequence of intermediate probability distributions: $p_0, ..., p_S$, with $p_0 = p_A$ and $p_S = p_B$. One general way to define this sequence is to set:

$$p_k(\mathbf{x}) \propto p_A^*(\mathbf{x})^{1-\beta_k} p_B^*(\mathbf{x})^{\beta_k}, \tag{8}$$

with "inverse temperatures" $0 = \beta_0 < \beta_1 < ... < \beta_K = 1$ chosen by the user. For each intermediate distribution, a Markov chain transition operator $T_k(\mathbf{x}'; \mathbf{x})$ that leaves $p_k(\mathbf{x})$ invariant must also be defined.

Using the special bipartite structure of RBM's, we can devise a better AIS scheme [14] for estimating the model's partition function. Let us consider a Replicated Softmax model with $D$ words. Using Eq. 5, the joint distribution over $\{\mathbf{V}, \mathbf{h}\}$ is defined as[1]:

$$p(\mathbf{V}, \mathbf{h}) = \frac{1}{\mathcal{Z}} \exp\left(\sum_{j=1}^{F} \sum_{k=1}^{K} W_j^k h_j \hat{v}^k\right), \tag{9}$$

where $\hat{v}^k = \sum_{i=1}^{D} v_i^k$ denotes the count for the $k^{th}$ word. By explicitly summing out the latent topic units $\mathbf{h}$ we can easily evaluate an unnormalized probability $p^*(\mathbf{V})$. The sequence of intermediate distributions, parameterized by $\beta$, can now be defined as follows:

$$p_s(\mathbf{V}) = \frac{1}{\mathcal{Z}_s} p^*(\mathbf{V}) = \frac{1}{\mathcal{Z}_s} \sum_{\mathbf{h}} p_s^*(\mathbf{V}, \mathbf{h}) = \frac{1}{\mathcal{Z}_s} \prod_{j=1}^{F}\left(1 + \exp\left(\beta_s \sum_{k=1}^{K} W_j^k \hat{v}^k\right)\right). \tag{10}$$

Note that for $s = 0$, we have $\beta_s = 0$, and so $p_0$ represents a uniform distribution, whose partition function evaluates to $\mathcal{Z}_0 = 2^F$, where $F$ is the number of hidden units. Similarly, when $s = S$, we have $\beta_s = 1$, and so $p_S$ represents the distribution defined by the Replicated Softmax model. For the intermediate values of $s$, we will have some interpolation between uniform and target distributions. Using Eqs. 3, 4, it is also straightforward to derive an efficient Gibbs transition operator that leaves $p_s(\mathbf{V})$ invariant.

A single run of AIS procedure is summarized in Algorithm 1. It starts by first sampling from a simple uniform distribution $p_0(\mathbf{V})$ and then applying a series of transition operators $T_1, T_2, \ldots, T_{S-1}$ that "move" the sample through the intermediate distributions $p_s(\mathbf{V})$ towards the target distribution $p_S(\mathbf{V})$. Note that there is no need to compute the normalizing constants of any intermediate distributions. After performing $M$ runs of AIS, the importance weights $w_{\text{AIS}}^{(i)}$ can be used to obtain an unbiased estimate of our model's partition function $\mathcal{Z}_S$:

$$\frac{\mathcal{Z}_S}{\mathcal{Z}_0} \approx \frac{1}{M} \sum_{i=1}^{M} w_{\text{AIS}}^{(i)}, \tag{11}$$

where $\mathcal{Z}_0 = 2^F$. Observe that the Markov transition operators do not necessarily need to be ergodic. In particular, if we were to choose dumb transition operators that do nothing, $T_s(\mathbf{V}' \leftarrow \mathbf{V}) = \delta(\mathbf{V}' - \mathbf{V})$ for all $s$, we simply recover the simple importance sampling procedure of Eq. 7.

When evaluating the probability of a collection of several documents, we need to perform a separate AIS run per document, if those documents are of different lengths. This is because each different-sized document can be represented as a separate RBM that has its own global normalizing constant.

## 4   Experimental Results

In this section we present experimental results on three three text datasets: NIPS proceedings papers, 20-newsgroups, and Reuters Corpus Volume I (RCV1-v2) [10], and report generalization performance of Replicated Softmax and LDA models.

### 4.1   Description of Datasets

The NIPS proceedings papers[2] contains 1740 NIPS papers. We used the first 1690 documents as training data and the remaining 50 documents as test. The dataset was already preprocessed, where each document was represented as a vector containing 13,649 word counts.

The 20-newsgroups corpus contains 18,845 postings taken from the Usenet newsgroup collection. The corpus is partitioned fairly evenly into 20 different newsgroups, each corresponding to a separate topic.[3] The data was split by date into 11,314 training and 7,531 test articles, so the training and test sets were separated in time. We further preprocessed the data by removing common stopwords, stemming, and then only considering the 2000 most frequent words in the training dataset. As a result, each posting was represented as a vector containing 2000 word counts. No other preprocessing was done.

The Reuters Corpus Volume I is an archive of 804,414 newswire stories[4] that have been manually categorized into 103 topics. The topic classes form a tree which is typically of depth 3. For this dataset, we define the relevance of one document to another to be the fraction of the topic labels that agree on the two paths from the root to the two documents. The data was randomly split into 794,414 training and 10,000 test articles. The available data was already in the preprocessed format, where common stopwords were removed and all documents were stemmed. We again only considered the 10,000 most frequent words in the training dataset.

For all datasets, each word count $w_i$ was replaced by $\log(1 + w_i)$, rounded to the nearest integer, which slightly improved retrieval performance of both models. Table 1 shows description of all three datasets.

### 4.2   Details of Training

For the Replicated Softmax model, to speed-up learning, we subdivided datasets into minibatches, each containing 100 training cases, and updated the parameters after each minibatch. Learning was carried out using Contrastive Divergence by starting with one full Gibbs step and gradually increaing to five steps during the course of training, as described in [14]. For all three datasets, the total number of parameter updates was set to 100,000, which took several hours to train. For the

| Data set | Number of docs | | $K$ | $\bar{D}$ | St. Dev. | Avg. Test perplexity per word (in nats) | | | |
|---|---|---|---|---|---|---|---|---|---|
| | Train | Test | | | | LDA-50 | LDA-200 | R. Soft-50 | Unigram |
| NIPS | 1,690 | 50 | 13,649 | 98.0 | 245.3 | 3576 | 3391 | 3405 | 4385 |
| 20-news | 11,314 | 7,531 | 2,000 | 51.8 | 70.8 | 1091 | 1058 | 953 | 1335 |
| Reuters | 794,414 | 10,000 | 10,000 | 94.6 | 69.3 | 1437 | 1142 | 988 | 2208 |

Table 1: Results for LDA using 50 and 200 topics, and Replaced Softmax model that uses 50 topics. $K$ is the vocabulary size, $\bar{D}$ is the mean document length, St. Dev. is the estimated standard deviation in document length.

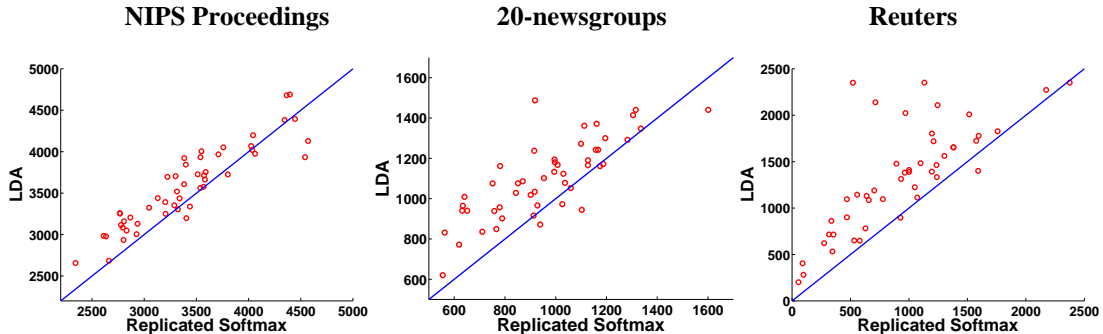

Figure 2: The average test perplexity scores for each of the 50 held-out documents under the learned 50-dimensional Replicated Softmax and LDA that uses 50 topics.

LDA model, we used the Gibbs sampling implementation of the Matlab Topic Modeling Toolbox[5] [5]. The hyperparameters were optimized using stochastic EM as described by [15]. For the 20-newsgroups and NIPS datasets, the number of Gibbs updates was set to 100,000. For the large Reuters dataset, it was set to 10,000, which took several days to train.

## 4.3 Assessing Topic Models as Generative Models

For each of the three datasets, we estimated the log-probability for 50 held-out documents.[6] For both the Replicated Softmax and LDA models we used 10,000 inverse temperatures $\beta_s$, spaced uniformly from 0 to 1. For each held-out document, the estimates were averaged over 100 AIS runs. The average test perplexity per word was then estimated as $\exp\left(-1/N \sum_{n=1}^{N} 1/D_n \log p(\mathbf{v}_n)\right)$, where $N$ is the total number of documents, $D_n$ and $\mathbf{v}_n$ are the total number of words and the observed word-count vector for a document $n$.

Table 1 shows that for all three datasets the 50-dimensional Replicated Softmax consistently outperforms the LDA with 50-topics. For the NIPS dataset, the undirected model achieves the average test perplexity of 3405, improving upon LDA's perplexity of 3576. The LDA with 200 topics performed much better on this dataset compared to the LDA-50, but its performance only slightly improved upon the 50-dimensional Replicated Softmax model. For the 20-newsgroups dataset, even with 200 topics, the LDA could not match the perplexity of the Replicated Softmax model with 50 topic units.

The difference in performance is particularly striking for the large Reuters dataset, whose vocabulary size is 10,000. LDA achieves an average test perplexity of 1437, substantially reducing it from 2208, achieved by a simple smoothed unigram model. The Replicated Softmax further reduces the perplexity down to 986, which is comparable in magnitude to the improvement produced by the LDA over the unigram model. LDA with 200 topics does improve upon LDA-50, achieving a perplexity of 1142. However, its performance is still considerably worse than that of the Replicated Softmax model.

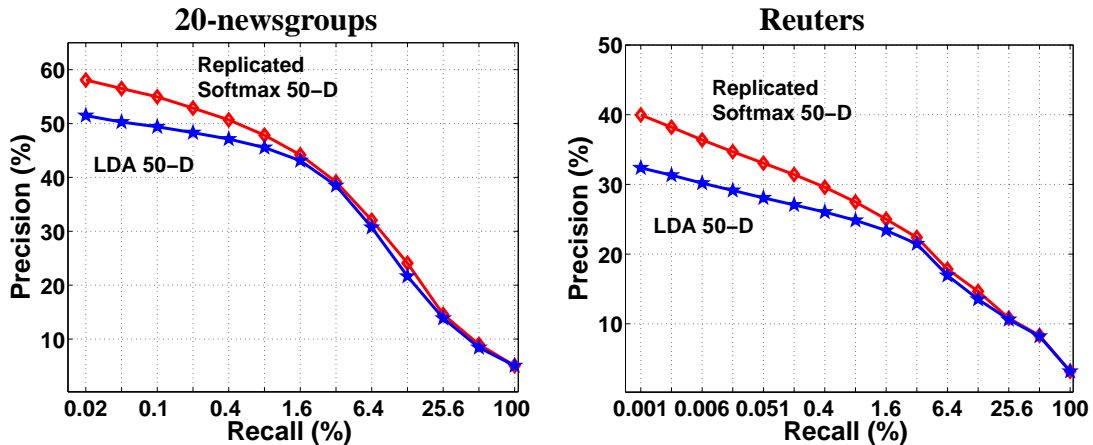

Figure 3: Precision-Recall curves for the 20-newsgroups and Reuters datasets, when a query document from the test set is used to retrieve similar documents from the training corpus. Results are averaged over all 7,531 (for 20-newsgroups) and 10,000 (for Reuters) possible queries.

Figure 2 further shows three scatter plots of the average test perplexity per document. Observe that for almost all test documents, the Replicated Softmax achieves a better perplexity compared to the corresponding LDA model. For the Reuters dataset, as expected, there are many documents that are modeled much better by the undirected model than an LDA. Clearly, the Replicated Softmax is able to generalize much better.

### 4.4 Document Retrieval

We used 20-newsgroup and Reuters datasets to evaluate model performance on a document retrieval task. To decide whether a retrieved document is relevant to the query document, we simply check if they have the same class label. This is the only time that the class labels are used. For the Replicated Softmax, the mapping from a word-count vector to the values of the latent topic features is fast, requiring only a single matrix multiplication followed by a componentwise sigmoid non-linearity. For the LDA, we used 1000 Gibbs sweeps per test document in order to get an approximate posterior over the topics. Figure 3 shows that when we use the cosine of the angle between two topic vectors to measure their similarity, the Replicated Softmax significantly outperforms LDA, particularly when retrieving the top few documents.

## 5 Conclusions and Extensions

We have presented a simple two-layer undirected topic model that be used to model and automatically extract distributed semantic representations from large collections of text corpora. The model can be viewed as a family of different-sized RBM's that share parameters. The proposed model have several key advantages: the learning is easy and stable, it can model documents of different lengths, and computing the posterior distribution over the latent topic values is easy. Furthermore, using stochastic gradient descent, scaling up learning to billions of documents would not be particularly difficult. This is in contrast to directed topic models, where most of the existing inference algorithms are designed to be run in a batch mode. Therefore one would have to make further approximations, for example by using particle filtering [3]. We have also demonstrated that the proposed model is able to generalize much better than LDA in terms of both the log-probability on held-out documents and the retrieval accuracy.

In this paper we have only considered the simplest possible topic model, but the proposed model can be extended in several ways. For example, similar to supervised LDA [1], the proposed Replicated Softmax can be easily extended to modeling the joint the distribution over words and a document label, as shown in Fig. 4, left panel. Recently, [11] introduced a Dirichlet-multinomial regression model, where a prior on the document-specific topic distributions was modeled as a function of observed metadata of the document. Similarly, we can define a conditional Replicated Softmax model, where the observed document-specific metadata, such as author, references, etc., can be used

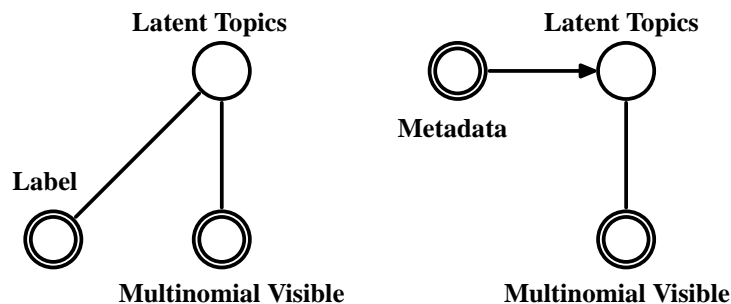

Figure 4: **Left:** A Replicated Softmax model that models the joint distribution of words and document label. **Right:** Conditional Replicated Softmax model where the observed document-specific metadata affects binary states of the hidden topic units.

to influence the states of the latent topic units, as shown in Fig. 4, right panel. Finally, as argued by [13], a single layer of binary features may not the best way to capture the complex structure in the count data. Once the Replicated Softmax has been trained, we can add more layers to create a Deep Belief Network [8], which could potentially produce a better generative model and further improve retrieval accuracy.

### Acknowledgments
This research was supported by NSERC, CFI, and CIFAR.

## Footnotes

[1]We have omitted the bias terms for clarity of presentation

[2] Available at http://psiexp.ss.uci.edu/research/programs_data/toolbox.htm.

[3] Available at http://people.csail.mit.edu/jrennie/20Newsgroups (20news-bydate.tar.gz).

[4] Available at http://trec.nist.gov/data/reuters/reuters.html

[5]The code is available at http://psiexp.ss.uci.edu/research/programs_data/toolbox.htm

[6]For the 20-newsgroups and Reuters datasets, the 50 held-out documents were randomly sampled from the test sets.

# References

[1] D. Blei and J. McAuliffe. Supervised topic models. In *NIPS*, 2007.

[2] D. Blei, A. Ng, and M. Jordan. Latent dirichlet allocation. *Journal of Machine Learning Research*, 3:993–1022, 2003.

[3] K. Canini, L. Shi, and T. Griffiths. Online inference of topics with latent Dirichlet allocation. In *Proceedings of the International Conference on Artificial Intelligence and Statistics*, volume 5, 2009.

[4] P. Gehler, A. Holub, and M. Welling. The Rate Adapting Poisson (RAP) model for information retrieval and object recognition. In *Proceedings of the 23rd International Conference on Machine Learning*, 2006.

[5] T. Griffiths and M. Steyvers. Finding scientific topics. In *Proceedings of the National Academy of Sciences*, volume 101, pages 5228–5235, 2004.

[6] Thomas Griffiths and Mark Steyvers. Finding scientific topics. *PNAS*, 101(suppl. 1), 2004.

[7] G. Hinton. Training products of experts by minimizing contrastive divergence. *Neural Computation*, 14(8):1711–1800, 2002.

[8] G. Hinton, S. Osindero, and Y. W. Teh. A fast learning algorithm for deep belief nets. *Neural Computation*, 18(7):1527–1554, 2006.

[9] T. Hofmann. Probabilistic latent semantic analysis. In *Proceedings of the 15th Conference on Uncertainty in AI*, pages 289–296, San Fransisco, California, 1999. Morgan Kaufmann.

[10] D. Lewis, Y. Yang, T. Rose, and F. Li. RCV1: A new benchmark collection for text categorization research. *Journal of Machine Learning Research*, 5:361–397, 2004.

[11] D. Mimno and A. McCallum. Topic models conditioned on arbitrary features with dirichlet-multinomial regression. In *UAI*, pages 411–418, 2008.

[12] R. Neal. Annealed importance sampling. *Statistics and Computing*, 11:125–139, 2001.

[13] R. Salakhutdinov and G. Hinton. Semantic Hashing. In *SIGIR workshop on Information Retrieval and applications of Graphical Models*, 2007.

[14] R. Salakhutdinov and I. Murray. On the quantitative analysis of deep belief networks. In *Proceedings of the International Conference on Machine Learning*, volume 25, pages 872 – 879, 2008.

[15] H. Wallach. Topic modeling: beyond bag-of-words. In *ICML*, volume 148, pages 977–984, 2006.

[16] H. Wallach, I. Murray, R. Salakhutdinov, and D. Mimno. Evaluation methods for topic models. In *Proceedings of the 26th International Conference on Machine Learning (ICML 2009)*, 2009.

[17] E. Xing, R. Yan, and A. Hauptmann. Mining associated text and images with dual-wing harmoniums. In *Proceedings of the 21st Conference on Uncertainty in Artificial Intelligence (UAI-2005)*, 2005.

